# PARTITIONING OF SENSORY DATA BY A CORTICAL NETWORK[1]

Richard Granger, José Ambros-Ingerson, Howard Henry, Gary Lynch
Center for the Neurobiology of Learning and Memory
University of California
Irvine, CA. 91717

## SUMMARY

To process sensory data, sensory brain areas must preserve information about both the similarities and differences among learned cues: without the latter, acuity would be lost, whereas without the former, degraded versions of a cue would be erroneously thought to be distinct cues, and would not be recognized. We have constructed a model of piriform cortex incorporating a large number of biophysical, anatomical and physiological parameters, such as two-step excitatory firing thresholds, necessary and sufficient conditions for long-term potentiation (LTP) of synapses, three distinct types of inhibitory currents (short IPSPs, long hyperpolarizing currents (LHP) and long cell-specific afterhyperpolarization (AHP)), sparse connectivity between bulb and layer-II cortex, caudally-flowing excitatory collateral fibers, nonlinear dendritic summation, etc. We have tested the model for its ability to learn similarity- and difference-preserving encodings of incoming sensory cues; the biological characteristics of the model enable it to produce multiple encodings of each input cue in such a way that different readouts of the cell firing activity of the model preserve both similarity and difference information.

In particular, probabilistic quantal transmitter-release properties of piriform synapses give rise to probabilistic postsynaptic voltage levels which, in combination with the activity of local patches of inhibitory interneurons in layer II, differentially select bursting vs. single-pulsing layer-II cells. Time-locked firing to the theta rhythm (Larson and Lynch, 1986) enables distinct spatial patterns to be read out against a relatively quiescent background firing rate. Training trials using the physiological rules for induction of LTP yield stable layer-II-cell spatial firing patterns for learned cues. Multiple simulated olfactory input patterns (i.e., those that share many chemical features) will give rise to strongly-overlapping bulb firing patterns, activating many shared lateral olfactory tract (LOT) axons innervating layer Ia of piriform cortex, which in turn yields highly overlapping layer-II-cell excitatory potentials, enabling this spatial layer-II-cell encoding to preserve the overlap (similarity) among similar inputs. At the same time, those synapses that are enhanced by the learning process cause stronger cell firing, yielding strong, cell-specific afterhyperpolarizing (AHP) currents. Local inhibitory interneurons effectively select alternate cells to fire once strongly-firing cells have undergone AHP. These alternate cells then activate their caudally-flowing recurrent collaterals, activating distinct populations of synapses in caudal layer Ib. Potentiation of these synapses in combination with those of still-active LOT axons selectively enhance the response of caudal cells that tend to accentuate the differences among even very-similar cues.

Empirical tests of the computer simulation have shown that, after training, the initial spatial layer II cell firing responses to similar cues enhance the similarity of the cues, such that the overlap in response is equal to or greater than the overlap in

input cell firing (in the bulb): e.g., two cues that overlap by 65% give rise to response patterns that overlap by 80% or more. Reciprocally, later cell firing patterns (after AHP), increasingly enhance the differences among even very-similar patterns, so that cues with 90% input overlap give rise to output responses that overlap by less than 10%. This difference-enhancing response can be measured with respect to its acuity; since 90% input overlaps are reduced to near zero response overlaps, it enables the structure to distinguish between even very-similar cues. On the other hand, the similarity-enhancing response is properly viewed as a *partitioning* mechanism, mapping quite-distinct input cues onto nearly-identical response patterns (or category indicators). We therefore use a statistical metric for the information value of categorizations to measure the value of partitionings produced by the piriform simulation network.

## INTRODUCTION

The three primary dimensions along which network processing models vary are their learning rules, their performance rules and their architectural structures. In practice, performance rules are much the same across different models, usually being some variant of a 'weighted-sum' rule (in which a unit's output is calculated as some function of the sum of its inputs multiplied by their 'synaptic' weights). Performance rules are usually either 'static' rules (calculating unit outputs and halting) or 'settling' rules (iteratively calculating outputs until a convergent solution is reached). Most learning rules are either variants of a 'correlation' rule, loosely based on Hebb's (1949) postulate; or a 'delta' rule, e.g., the perceptron rule (Rosenblatt, 1962), the adaline rule (Widrow and Hoff, 1960) or the generalized delta or 'backpropagation' rule (Parker, 1985; Rumelhart et al., 1986). Finally, architectures vary by and large with learning rules: e.g., multi-layered feedforward nets require a generalized delta rule for convergence; bidirectional connections usually imply a variant of a Hebbian or correlation rule, etc.

Architectures and learning and performance rules are typically arrived at for reasons of their convenient computational properties and analytical tractability. These rules are sometimes based in part on some results borrowed from neurobiology: e.g., 'units' in some network models are intended to correspond loosely to neurons, and 'weights' loosely to synapses; the notions of parallelism and distributed processing are based on metaphors derived from neural processes.

An open question is how much of the rest of the rich literature of neurobiological results should or could profitably be incorporated into a network model. From the point of view of constructing mechanisms to perform certain pre-specified computatonal functions (e.g., correlation, optimization), there are varying answers to this question. However, the goal of understanding brain circuit function introduces a fundamental problem: there are no known, pre-specified functions of any given cortical structures. We have constructed and studied a physiologically- and anatomically-accurate model of a particular brain structure, olfactory cortex, that is strictly based on biological data, with the goal of elucidating the local function of this circuit from its performance in a 'bottom-up' fashion. We measure our progress by the accuracy with which the model corresponds to known data, and predicts novel physiological results (see, e.g., Lynch and Granger, 1988; Lynch et al., 1988).

Our initial analysis of the circuit reveals a mechanism consisting of a learning rule that is notably simple and restricted compared to most network models, a relatively novel architecture with some unusual properties, and a performance rule that is ex-

traordinarily complex compared to typical network-model performance rules. Taken together, these rules, derived directly from the known biology of the olfactory cortex, generate a coherent mechanism that has interesting computational properties. This paper describes the learning and performance rules and the architecture of the model; the relevant physiology and anatomy underlying these rules and structures, respectively; and an analysis of the coherent mechanism that results.

## LEARNING RULES DERIVED FROM LONG-TERM POTENTIATION

Long-term potentiation (LTP) of synapses is a phenomenon in which a brief series of biochemical events gives rise to an enhancement of synaptic efficacy that is extraordinarily long-lasting (Bliss and Lømo, 1973; Lynch and Baudry, 1984; Staubli and Lynch, 1987); it is therefore a candidate mechanism underlying certain forms of learning, in which few training trials are required for long-lasting memory. The physiological characteristics of LTP form the basis for a straightforward network learning rule.

It is known that simultaneous pre- and post-synaptic activity (i.e., intense depolarization) result in LTP (e.g., Wigstrøm et al., 1986). Since excitatory cells are embedded in a meshwork of inhibitory interneurons, the requisite induction of adequate levels of pre- and postsynaptic activity is achieved by stimulation of large numbers of afferents for prolonged periods, by voltage clamping the postsynaptic cell, or by chemically blocking the activity of inhibitory interneurons. In the intact animal, however, the question of how simultaneous pre- and postsynaptic activity might be induced has been an open question. Recent work (Larson and Lynch, 1986) has shown that when hippocampal afferents are subjected to patterned stimulation with particular temporal and frequency parameters, inhibition is naturally eliminated within a specific time window, and LTP can arise as a result. Figure 1 shows that LTP naturally occurs using short (3-4 pulse) bursts of high-frequency (100Hz) stimulation with a 200ms interburst interval; only the second of a pair of two such bursts causes potentiation. This occurs because the normal short inhibitory currents (IPSPs), which prevent the first burst from depolarizing the postsynaptic cell sufficiently to produce LTP, are maximally refractory at 200ms after being stimulated, and therefore, although the second burst arrives against a hyperpolarized background resulting from the long hyperpolarizing currents (LHP) initiated by the first burst, the second burst does not initiate its own IPSPs, since they are then refractory. The studies leading to these conclusions were performed in *in vitro* hippocampal slices; LTP induced by this patterned stimulation technique in intact animals shows no measurable decrement prior to the time at which recording arrangements deteriorate: more than a month in some cases (see Staubli and Lynch, 1987).

## PERFORMANCE RULES DERIVED FROM OLFACTORY PHYSIOLOGY AND BEHAVIOR

From the above data we may infer that LTP itself depends on simultaneous pre- and postsynaptic activity, as Hebb postulated, but that a sufficient degree of the latter occurs only under particular conditions. Those conditions (patterned stimulation) suggest the beginnings of a performance rule for the network. Drawing this out requires a review of the inhibitory currents active in hippocampus and in piriform cortex. Three classes of such currents are known to be present: short IPSPs, long LHPs and extremely long, cell-specific afterhyperpolarization, or AHP (see Figure 2). Short IPSPs arise from both feedforward and feedback activation of inhibitory interneurons which in turn synapse

on excitatory cells (e.g., layer II cells, which are primary excitatory cells in piriform). IPSPs develop more slowly than excitatory postsynaptic potentials (EPSPs) but quickly shunt the EPSP, thus reversing the depolarization that arises from EPSPs, and bringing the cell voltage down below its original resting potential. IPSPs last approximately 50–100ms, and then enter a refractory period during which they cannot be reactivated from about 100–300ms after they have been once activated. Longer hyperpolarization (LHP) is presumably dependent on a distinct type of inhibitory interneuron or inhibitory receptor, and arises in much the same way; however, these cells are apparently not refractory once activated. LHP lasts for 300-500ms.

Taken together, IPSPs and LHP constitute a form of high-pass frequency filter: 200ms after an input burst, a subsequent input will arrive against a background of hyperpolarization due to LHP, yet this input will not initiate its own IPSP due to the refractory period. If the input is a single pulse, its EPSP will fail to trigger the postsynaptic cell, since it will not be able to overcome the LHP-induced hyperpolarized potential of the cell. Yet if the input is a high-frequency burst, the pulses comprising the burst will give rise to different behavior. Ordinarily, the first EPSP would have been driven back to resting potential by its accompanying IPSP, before the second pulse in the burst could arrive. But when the IPSP is absent, the first EPSP is not driven rapidly down to resting potential, and the second pulse sums with it, raising the voltage of the postsynaptic cell and allowing voltage-dependent channels to open, thereby further depolarizing the cell, and causing it to spike (Figure 3). Hence these high-frequency bursts fire the cell, while single pulses or lower-frequency bursts would not do so. When these cells fire, then active synapses can be potentiated.

The third inhibitory mechanism, AHP, is a current that causes an excitatory cell to become refractory after it has fired strongly or rapidly. This mechanism is therefore specific to those cells that have fired, unlike the first two mechanisms. AHP can prevent a cell from firing again for as long as 1000ms (1 second).

It has long been observed that EEG waves in the hippocampi of learning animals are dominated by the theta rhythm, i.e., activity occuring at about 4-8Hz. This is now seen to correspond to the optimal rate for firing postsynaptic cells and for enhancing synapses via LTP; i.e., this rhythmic aspect of the performance rules of these networks is suggested by the physiology of LTP. The resulting activation patterns may take the following form: relatively synchronized cell firing occurring approximately once every 200ms, i.e., spatial patterns of induced activity occurring at the rate of one new spatial cell-firing pattern every 200ms. The cells most strongly participating in any one firing pattern will not participate in subsequent patterns (at least the next 4-5 patterns, i.e., 800-1000ms), due to AHP. This raises the interesting possibility that different spatial patterns (at different times) may be conveying different information about their inputs. In summary, postsynaptic cells fire in pulses or bursts depending on the synaptically-weighted sums of their active axonal inputs; this firing is synchronized across the cells in a structure, giving rise to a spatial pattern of activity across these cells; once cells fire they will not fire again in subsequent patterns; each pattern (occuring at the theta rhythm, i.e., approximately once every 200ms) will therefore consist of extremely different spatial patterns of cell activity. Hence the 'output' of such a network is a sequence of spatial patterns.

In an animal engaged in an olfactory discrimination learning task, the theta rhythm

dominates the animals behavior: the animals literally sniff at theta. We have been able to sustitute direct stimulation (in theta-burst mode) of the lateral olfactory tract (LOT), which is the input to the olfactory cortex, for odors: these 'electrical odors' are learned and discriminated by the animals, either from other electrical odors (via different stimulating electrodes) or from real odors. Furthermore, behavioral learning in this paradigm is accompanied by LTP of piriform synapses (Roman et al., 1987). This experimental paradigm thus provides us with a known set of behaviorally-relevant inputs to the olfactory cortex that give rise to synaptic potentiation that apparently underlies the learning of the stimuli.

## ARCHITECTURE OF OLFACTORY CORTEX

Nasal receptor cells respond differentially to different chemicals; these cells topographically innervate the olfactory bulb, which is arranged such that combinations of specific spatial 'patches' of bulb characteristically respond to specific odors. Bulb also receives a number of centrifugal afferents from brain, most of which terminate on the inhibitory granule cells. The excitatory mitral cells in bulb send out axons that form the lateral olfactory tract (LOT), which constitutes the only major input to olfactory (piriform) cortex. This cortex in turn has some feedback connections to bulb via the anterior olfactory nucleus.

Figure 4 illustrates the anatomy of the superficial layers of olfactory cortex: the LOT axons flow across layer Ia, synapsing with the dendrites of piriform layer-II cells. Those cells in turn give rise to collateral axon outputs which flow, in layer Ib, parallel and subjacent to the LOT, in a predominantly rostral-to-caudal direction, eventually terminating in entorhinal cortex. Layer Ia is very sparsely connected; the probability of synapses between LOT axons and layer-II cell dendrites is less than 0.10 (Lynch, 1986), and decreases caudally. Layer Ib (where collaterals synapse with dendrites) is also sparse, but its density increases caudally, as the number of collaterals increases; the overall connectivity density on layer-II-cell dendrites is approximately constant throughout most of piriform. Layer II also contains, in addition to the principal excitatory cells (modified stellates), inhibitory interneurons which synapse on excitatory cells within a specified radius, forming a 'patchwork' of cells affected by a particular inhibitory cell; the spheres of influence of inhibitory cells almost certainly overlap somewhat. There are approximately 50,000 LOT axons, 500,000 piriform layer II cells, and a much smaller number of inhibitory cells that divide layer II roughly into functional patches. (See Price, 1973; Luskin and Price, 1983; Krettek and Price, 1977; Price and Slotnick, 1983; Haberly and Price, 1977, 1978a, 1978b).

The layer II cell collateral axons flow through layer III for a distance before rising up to layer Ib (Haberly, 1985); taken in combination with the predominantly caudal directionality of these collaterals, this means that rostral piriform will be dominated by LOT inputs. Extreme caudal piriform (and all of lateral entorhinal cortex) is dominated by collaterals from more rostral cells; moving from rostral to caudal piriform, cells increasingly can be thought of as 'hybrid cells': cells receiving inputs from both the bulb (via the LOT) and from rostral piriform (via collateral axons). The architectural characteristics of rostral piriform is therefore quite different from that of caudal piriform, and differential analysis must be performed of rostral cells vs. hybrid cells, as will be seen later in the paper.

## SIMULATION AND FORMAL ANALYSIS: INTRODUCTION

We have conducted several simulations of olfactory cortex incorporating many of the physiological features discussed earlier. Two hundred layer II cells are used with 100 input (LOT) lines and 200 collateral axons; both the LOT and collateral axons flow caudally. LOT axons connect with rostral dendrites with a probability of 0.2, which decreases linearly to 0.05 by the caudal end of the model. The connectivity is arranged randomly, subject to the constraint that the number of contacts for axons and dendrites is fixed within certain narrow boundaries (in the most severe case, each axon forms 20 synapses and each dendrite receives 20 contacts). The resulting matrix is thus hypergeometric in both dimensions. There are 20 simulated inhibitory interneurons, such that the layer II cells are arranged in 20 overlapping patches, each within the influence of one such inhibitory cell. Inhibition rules are approximately as discussed above; i.e., the short IPSP is longer than an EPSP but only one fifth the length of the LHP; cell-specific AHP in turn is twice as long as LHP.

Synaptic activity in the model is probabilistic and quantal: for any presynaptic activation, there is a fixed probability that the synapse will allow a certain amount of conductance to be contributed to the postsynaptic cell. Long-term potentiation was represented by a 40% increase in contact strength, as well as an increase in the probability of conductance being transmitted. These effects would be expected to arise, *in situ*, from modifying existing synapses as well as adding new ones (Lynch, 1986), two results obtained in electron microscopic studies (Lee et al., 1980). Only excitatory cell synapses are subject to LTP. LTP occurred when a cell was activated twice at a simulated 200ms interval: the first input 'primes' the synapse so that a subsequent burst input can drive it past a threshold value; following from the physiological results, previously potentiated synapses were much less different from "naive" synapses when driven at high frequency (see Lynch et al., 1988). The simulation used theta burst activation (i.e., bursts of pulses with the bursts occurring at 5Hz) of inputs during learning, and operated according to these synchronized fixed time steps, as discussed above.

The network was trained on sets of "odors", each of which was represented as a group of active LOT lines, as in the "electric odor" experiments already described. Usually three or four "components" were used in an odor, with each component consisting of a group of contiguous LOT lines. We assumed that the bulb normalized the output signal to about 20% of all LOT fibers. In some cases, more specific bulb rules were used and in particular inhibition was assumed to be greatest in areas surrounding an active bulb "patch".

The network exhibited several interesting behaviors. Learning, as expected, increased the robustness of the response to specific vectors; thus adding or subtracting LOT lines from a previously learned input did not, within limits, greatly change the response. The model, like most network simulations, dealt reasonably well with degraded or noisy known signals. An unexpected result developed after the network had learned a succession of cues. In experiments of this type, the simulation would begin to generate two quite distinct output signals within a given sampling episode; that is, a single previously learned cue would generate two successive responses in successive 'sniffs' presented to an "experienced" network. The first of these response patterns proved to be common to several signals while the second was specific to each learned signal. The

common signal was found to occur when the network had learned 3–5 inputs which had substantial overlap in their components (e.g., four odors that shared ≈70% of their components). It appeared then that the network had begun to produce "category" or "clustering" responses, on the first sniff of a simulated odor, and "individual" or "differentiation" responses on subsequent sniffs of that same odor. When presented with a novel cue which contained elements shared with other, previously learned signals, the network produced the cluster response but no subsequent individual or specific output signal. Four to five cluster response patterns and 20 – 25 individual responses were produced in the network without distortion.

In retrospect, it was clear that the model accomplished two necessary and in some senses opposing operations: 1) it detected similarities in the members of a cue category or cluster, and, 2) it nonetheless distinguished between cues that were quite similar. Its first response was to the similarity-based category and its second to the specific signal.

## ANALYSIS OF CATEGORIZATION IN ROSTRAL PIRIFORM

Assume that a set of input cues (or 'simulated odors') $X^\alpha, X^\beta \ldots X^\zeta$ differ from each other in the firing of $d_X$ LOT input lines; similarly, inputs $Y^\alpha, Y^\beta \ldots Y^\zeta$ differ in $d_Y$ lines, but that inputs from the sets $X$ and $Y$ differ from each other in $D_{X,Y} >> d$ lines, such that the $X$s and the $Y$s form distinct natural categories. Then the performance of the network should give rise to output (layer II cell) firing patterns that are very similar among members of either category, but different for members of different categories; i.e., there should be a single spatial pattern of response for members of $X$, with little variation in response across members, and there should be a distinct spatial pattern of response for members of $Y$.

Considering a matrix constructed by uniform selection of neurons, each with a hypergeometric distribution for its synapses, as an approximation of the bidimensional hypergeometric matrix described above, the following results can be derived. The expected value of $\hat{d}$, the Hamming distance between responses for two input cues differing by $2d$ LOT lines (input Hamming distance of $d$) is:

$$E(\hat{d}) = \sum_{k=1}^{N_o} \left[ \sum_{\substack{i > \theta \\ j < \theta}} S_i I(i,j) + \sum_{\substack{i < \theta \\ j \geq \theta}} S_i I(i,j) \right]$$

where $N_o$ is the number of postsynaptic cells, each $S_i$ is the probability that a cell will have precisely $i$ active contacts from one of the two cues, and $I(i,j)$ is the probability that the number of contacts on the cell will increase (or decrease) from $i$ to $j$ with the change in $d$ LOT lines; i.e., changing from the first cue to the second. Hence, the first term denotes the probability of a cell decreasing its number of active contacts from above to below some threshold, $\theta$, such that that cell fired in response to one cue but not the other (and therefore is one of the cells that will contribute to the difference between responses to the two cues). Reciprocally, the second term is the probability that the cell increases its number of active synapses such that it is now over the threshold; this cell also will contribute to the difference in response. We restrict our analysis for now to rostral piriform, in which there are assumed to be few if any collateral axons. We will return to this issue in the next subsection.

The value for each $S_a$, the probability of $a$ active contacts on a cell, is a hypergeometric function, since there are a fixed number of contacts anatomically between LOT and (rostral) piriform cells:

$$S_a = p(a\ active\ synapses) = \frac{\binom{A}{a}\binom{N-A}{n-a}}{\binom{N}{n}}$$

where $N$ is the number of LOT lines, $A$ is the number of active (firing) LOT lines, $n$ is the number of synapses per dendrite formed by the LOT, and $a$ is the number of active such synapses. The formula can be read by noting that the first binomial indicates the number of ways of choosing $a$ active synapses on the dendrite from the $A$ active incoming LOT lines; for each of these, the next expression calculates the number of ways in which the remaining $n - a$ (inactive) synapses on the dendrite are chosen from the $N - A$ inactive incoming LOT lines; the probability of active synapses on a dendrite depends on the sparseness of the matrix (i.e., the probability of connection between any given LOT line and dendrite); the solution must be normalized by the number of ways in which $n$ synapses on a dendrite can be chosen from $N$ incoming LOT lines.

The probability of a cell changing its number of contacts from $a$ to $\hat{a}$ is:

$$I(a, \hat{a}) = \sum_{\substack{g-l= \\ a-\hat{a}}} \left[ \frac{\binom{a}{l}\binom{A-a}{d-l}}{\binom{A}{d}} \frac{\binom{n-a}{g}\binom{N-A-(n-a)}{d-g}}{\binom{N-A}{d}} \right]$$

where $N$, $n$, $A$, and $a$ are as above, $l$ is the "loss" or reduction in the number of active synapses, and $g$ is the gain or increase. Hence the left expression is the probability of losing $l$ active synapses by changing $d$ LOT lines, and the right-hand expression is the probability of gaining $g$ active synapses. The product of the expressions are summed over all the ways of choosing $l$ and $g$ such that the net change $g - l$ is the desired difference $a - \hat{a}$.

If training on each cue induces only fractional LTP, then over trials, synapses contacted by any overlapping parts of the input cues should become stronger than those contacted only by unique parts of the cue. Comparing two cues from within a category, vs. two cues from between categories, there may be the same number of active synapses lost across the two cues in either case, but the expected *strength* of the synapses lost in the former case (within category) should be significantly lower than in the latter case (across categories). Hence, for a given threshold, the difference $\hat{d}$ between output firing patterns will be smaller for two within-category cues than for cues from two different categories.

It is important to note that clustering is an operation that is quite distinct from stimulus generalization. Observing that an object is a car does not occur because of a comparison with a specific, previously learned car. Instead the category "car" emerges from the learning of many different cars and may be based on a "prototype" that has no necessary correspondence with a specific, real object. The same could be said of the network. It did not produce a categorical response when one cue had been learned

and second similar stimulus was presented. Category or cluster responses, as noted, required the learning of several exemplars of a similarity-based cluster. It is the process of extracting commonalities from the environment that defines clustering, not the simple noting of similarities between two cues.

An essential question in clustering concerns the location of the boundaries of a given group; i.e., what degree of similarity must a set of cues possess to be grouped together? This issue has been discussed from any number of theoretical positions (e.g., information theory); all these analyses incorporate the point that the breadth of a category must reflect the overall homogeneity or heterogeneity of the environment. In a world where things are quite similar, useful categories will necessarily be composed of objects with much in common. Suppose, for instance, that subjects were presented with a set of four distinct coffee cups of different colors, and asked later to recall the objects. The subjects might respond by listing the cups as a blue, red, yellow and green coffee cup, reflecting a relatively specific level of description in the hierarchy of objects that are coffee cups. In contrast, if presented with four different objects, a blue coffee cup, a drinking glass, a silver fork and a plastic spoon, the cup would be much more likely to be recalled as simply a cup, or a coffee cup, and rarely as a blue coffee cup; the specificity of encoding chosen depends on the overall heterogeneity of the environment. The categories formed by the simulation were quite appropriate when judged by an information theoretic measure, but how well it does across a wide range of possible worlds has not been addressed.

## ANALYSIS OF PROBLEMS ARISING FROM CAUDAL AXON FLOW

The anatomical feature of directed flow of collateral axons gives rise to an immediate problem in principle. In essence, the more rostral cells that fire in response to an input, the more active inputs there are from these cells to the caudal cells, via collateral axons, such that the probability of caudal cell firing increases precipitously with probability of rostral cell firing. Conversely, reducing the number of rostral cells from firing, either by reducing the number of active input LOT axons or by raising the layer II cell firing threshold, prevents sufficient input to the caudal cells to enable their probability of firing to be much above zero.

This problem can be stated formally, by making assumptions about the detailed nature of the connectivity of LOT and collateral axons in layer I as these axons proceed from rostral to caudal piriform. The probability of contact between LOT axons and layer-II-cell dendrites decreases caudally, as the number of collateral axons is increasing, given their rostral to caudal flow tendency. This situation is depicted in Figure 4. Assuming that probability of LOT contact tends to go to zero, we may adopt a labelling scheme for axons and synaptic contacts, as in the diagram, in which some combination of LOT axons ($x_k$) and collateral axons ($h_m$) contact any particular layer II cell dendrite ($h_n$), each of which is itself the source of an additional collateral axon flowing to cells more caudal than itself. Then the cell firing function for layer II cell $h_n$ is:

$$h_n = H(\sum_{m<n} h_m w_{nm} + \sum_{k\geq n} x_k w_{nk} - \theta)$$

where the $x_k$ denote LOT axon activity of those axons still with nonzero probability of contact for layer II cell $h_n$, the $h_m$ denote activity of layer II cells rostral of $h_n$, $\theta$ is

the cell firing threshold, $w_{nm}$ is the synaptic strength between axon m and dendrite n, and $H$ is the Heaviside step function, equal to 1 or 0 according to whether its argument is positive or negative. If we assume instead that probability of cell firing is a graded function rather than a step function, we may eliminate the $H$ step function and calculate the firing of the cell $(h_n)$ from its inputs $(h_{n,net})$ via the logistic:

$$h_{n,net} = \sum_{m<n} h_m w_{nm} + \sum_{k \geq n} x_k w_{nk}$$

$$h_n = \frac{1}{1 + e^{-(kh_{n,net} + \theta_n)}}$$

Then we may expand the expression for firing of cell $h_n$ as follows:

$$h_n = \left[ 1 + e^{-(\sum_{m<n} h_m w_{nm} + \sum_{k \geq n} x_k w_{nk} + \theta)} \right]^{-1}$$

By assuming a fixed firing threshold, and varying the number of active input LOT lines, the probability of cell firing can be examined. Numerical simulation of the above expressions across a range of LOT spatial activation patterns demonstrates that probability of cell firing remains near zero until a critical number of LOT lines are active, at which point the probability flips to close to 100% (Figure 5). This means that, for any given firing threshold, given fewer than a certain amount of LOT input, practically no piriform cells will fire, whereas a slight increase in the number of active LOT lines will mean that practically all piriform cells should fire.

This excruciating dependence of cell firing on amount of LOT input indicates that normalization of the size of the LOT input alone will be insufficient to stabilize the size of the layer II response; even slight variation of LOT activity in either direction has extreme consequences. A number of solutions are possible; in particular, the known local anatomy and physiology of layer II inhibitory interneurons provides a mechanism for controlling the amount of layer II response. As discussed, inhibitory interneurons give rise to both feedforward (activated by LOT input) and feedback (activated by collateral axons) activity; the influence of any particuar interneuron is limited anatomically to a relatively small radius around itself within layer II, and the influence of multiple interneurons probably overlap to some extent. Nonetheless, the 'sphere of influence' of a particular inhibitory interneuron can be viewed as a local patch in layer II, within which the number of active excitatory cells is in large measure controlled by the activity of the inhibitory cell in that patch. If a number of excitatory cells are firing with varying depolarization levels within a patch in layer II, activation of the inhibitory cells by the excitatory cells will tend to weaken those excitatory cells that are less depolarized than the most strongly-firing cell within the patch, leading to a competition in which only those cells firing most strongly within a patch will burst, and these cells will, via the interneuron, suppress multiple firing of other cells within the patch. Thus the patch takes on some of the characteristics of a 'winner-take-all' network (Feldman, 1982): only the most strongly firing cells will be able to overcome inhibition sufficiently to burst, some additional cells will pulse once and then be overwhelmed by inhibition, and the rest of the cells in the patch will be silent, even though that patch may be receiving a large amount of excitatory input via LOT and collateral axon activity in layer I.

## EMERGENT CATEGORIZATION BEHAVIOR IN THE MODEL

The probabilistic quantal transmitter-release properties of piriform synapses described above give rise to probabilistic levels of postsynaptic depolarization. This inherent randomness of cell firing, in combination with activity of local inhibitory patches in layer II, selects different sets of bursting and pulsing cells on different trials if no synaptic enhancement has taken place. The time-locked firing to the theta rhythm enables distinct spatial patterns of firing to be read out against a relatively quiescent background firing rate. Synaptic LTP enhances the conductances and alters the probabilistic nature of communication between a given axon and dendrite, which tends to overcome the randomness of the cell firing patterns in untrained cells, yielding a stable spatial pattern that will reliably appear in response to the same input in the future, and in fact will appear even in response to degraded or noisy versions of the input pattern. Furthermore, subsequent input patterns that differ in only minor respects from a learned LOT input pattern will contact many of the already-potentiated synapses from the original pattern, thereby tending to give rise to a very similar (and stable) output firing pattern. Thus as multiple cues sharing many overlapping LOT lines are learned, the layer II cell responses to each of these cues will strongly resemble the responses to the others. Hence, the response(s) behave as though simply labelling a *category* of very-similar cues; sufficiently different cues will give rise to quite-different category responses.

## EMERGENT DIFFERENTIATION BEHAVIOR IN THE MODEL

Potentiated synapses cause stronger depolarization and firing of those cells participating in a 'category' response to a learned cue. This increased depolarization causes strong, cell-specific afterhyperpolarization (AHP), effectively putting those cells into a relatively long-lasting ($\approx$ 1sec) refractory period that prevents them from firing in response to the next few sampling sniffs of the cue. Then the inhibitory 'winner-take-all' behavior within patches effectively selects alternate cells to fire, once these strongly-firing (learned) cells have undergone AHP. These alternates will be selected with some randomness, given the probabilistic release characteristics discussed above, since these cells will tend not to have potentiated synapses. These alternate cells then activate their caudally-flowing recurrent collaterals, activating distinct populations of synapses in caudal layer Ib. Potentiation of these synapses in combination with those of still-active LOT axons tends to 'recruit' stable subpopulations of caudal cells that are distinct for each simulated odor. They are distinct for each odor because first rostral cells are selected from the population of unpotentiated or weakly-potentiated cells (after the strongly potentiated cells have been removed via AHP); hence they will at first tend to be selected randomly. Then, of the caudal cells that receive some activation from the weakening caudal LOT lines, those that also receive collateral innervation from these semi-randomly selected rostrals will be those that will tend to fire most strongly, and hence to be potentiated.

The probability of a cell participating in the rostral semi-randomly selected groups for more than one odor (e.g., for two similar odors) is lower than the probability of cells being recruited by these two odors initially, since the population are those that receive not enough input from the LOT to have been recruited as a category cell and potentiated, yet receive enough input to fire as an alternate cell. The probability of any caudal cell then being recruited for more than one odor by these rostral cell collaterals

in combination with weakening caudal LOT lines is similarly low. The product of these two probabilities is of course lower still. Hence, the probability that any particular caudal cell potentiated as part of this process will participate in response to more than one odor is very low.

This means that, when sampling (sniffing), the first pattern of cell firing will indicate similarity among learned odors, causing AHP of those patterns; thus later sniffs will generate patterns of firing that tend to be quite different for different odors, even when those odors are very similar. Empirical tests of the simulation have shown that odors consisting of 90%-overlapping LOT firing patterns will give rise to overlaps of between 85% and 95% in their initial layer II spatial firing patterns, whereas these same cues give rise to layer II patterns that overlap by less than 20% on 2nd and 3rd sniffs. The spatio-temporal pattern of layer II firing over multiple samples thus can be taken as a strong differentiating mechanism for even very-similar cues, while the initial sniff-response for those cues will nonetheless give rise to a spatial firing pattern that indicates the similarity of sets of learned cues, and therefore their 'category membership' in the clustering sense.

## CLUSTERING

Incremental clustering of cues into similarity-based categories is a more subtle process than might be thought and while it is clear that the piriform simulation performs this function, we do not know how optimal its performance is in an information-theoretic sense, relative to some measure of the value or cost of information in the encoding. Building a categorical scheme is a non-monotonic, combinatorial problem: that is, each new item to be learned can have disproportionate effects on the existing scheme, and the number of potential categories (clusters) climbs factorially with the number of items to be categorized. Algorithmic solutions to problems of this type are computationally very expensive. Calculation of an ideal categorization scheme (with respect to particular cost measures in a performance task), using a hill-climbing algorithm derived from an information-theoretic measure of category value, applied to a problem involving 22 simulated odors, required more than 4 hours on a 68020-based processor. The simulation network reached the same answer as the game-theoretic program, but did so in seconds. It is worth mentioning again that the simulation did so while simultaneously learning unique encodings for the cues, as described above, which is itself a nontrivial task.

Humans, on at least some tasks, may carry out clustering by building initial clusters and then merging or splitting them as more cues are presented. Thus far, the networks do not pass through successive categorization schema. However, experiments on human categorization have almost exclusively involved situations in which all cues were presented in rapid succession and category membership is taught explicitly, rather than developed independently by the subject. Hence, it is not clear from the experimental literature whether or not stable clusters develop in this way from stimuli presented at widely spaced intervals with no category membership information given, which is the problem corresponding to that given the network (and that is likely common in nature). It will be of interest to test categorizing skills of rats learning successive olfactory discriminations over several days. Using appropriately selected stimuli, it should be possible to determine if stable clusters are constructed and whether merging and splitting occurs over trials.

Any useful clustering device must utilize information about the heterogenity of the stimulus world in setting the heterogeneity of individual categories. Heterogeneity of categories refers to the degree of similarity that is used to determine if cues are to be grouped together or not. Several network parameters will influence category size and we are exploring how these influence the individuation function; one particularly interesting possibility involves a shifting threshold function, an idea used with great success by Cooper in his work on visual cortex. The problems presented to the simulation thus far involve a totally naive system, one that has had no "developmental" history. We are currently exploring a model in which early experiences are not learned by the network but instead set parameters for later ("adult") learning episodes. The idea is that early experience determines the heterogenity of the stimulus world and imprints this on the network, not by specific changes in synaptic strengths, but in a more general fashion.

## CONCLUSIONS

Neurons have a nearly bewildering array of biophysical, chemical, electrophysiological and anatomical properties that control their behavior; an open question in neural network research is which of these properties need be incorporated into networks in order to simulate brain circuit function. The simulation described here incorporates an extreme amount of biological data, and in fact has given rise to novel physiological questions, which we have tested experimentally with results that are counterintuitive and previously unsuspected in the existing physiological literature (see, e.g., Lynch and Granger, 1988; Lynch et al., 1988). Incorporation of this mass of physiological parameters into the simulation gives rise to a coherent architecture and learning and performance rules, when interpreted in terms of computational function of the network, which generates a robust capability to encode multiple levels of information about learned stimuli. The coherence of the data in the model is useful in two ways: to provide a framework for understanding the purposes and interactions of many apparently-disparate biological properties of neurons, and to aid in the design of novel artificial network architectures inspired by biology, which may have useful computational functions.

It is instructive to note that neurons are capable of many possible biophysical functions, yet early results from chronic recording of cells from olfactory cortex *in animals actively engaged in learning many novel odors in an olfactory discrimination task* clearly shows a particular operating mode of this cortical structure when it is actively in use by the animal (Larson et al., unpublished data). The rats in this task are very familiar with the testing paradigm and exhibit very raid learning, with no difficulty in acquiring large numbers of discriminations. Sampling, detection and responding occur in fractions of a second, indicating that the utilization of recognition memories in the olfactory system can be a rapid operation; it is not surprising, then, that the odor-coded units so far encountered in our physiological experiments have rapid and stereotyped responses. Given the dense innervation of the olfactory bulb by the brain, it is possible that the type of spatial encoding that appears to be responsible for the preliminary results of these chronic experiments would not appear in animals that were not engaged in active sampling or were confronted with unfamiliar problems. That is, the operation of the olfactory cortex might be as dependent upon the behavioral 'state' and behavioral history of the rat as upon the actual odors presented to it. It will be of interest to compare the results from well-trained freely-moving animals with those obtained using more restrictive testing conditions.

The temporal properties of synaptic currents and afterpotentials, results from simulations and chronic recording studies, taken together, suggest two useful caveats for biological models:

- Cell firing in cortical structures (e.g., piriform, hippocampus and possibly neocortex) is linked to particular rhythms (theta in the case of piriform and hippocampus) during real learning behavior, and thus it is likely that the 'coding language' of these structures involves spatial cell firing patterns within a brief time window. This stands in contrast to other methods such as frequency coding that appears in other structures (such as peripheral sensory structures, e.g., retina and cochlea; see, e.g., Sivilotti et al., 1987).

- Temporal sequences of spatial patterns may encode different types of information, such as hierarchical encodings of perceptions, in contrast with views in which either asynchronous 'cycling' activity occurs or a system yields a single punctate output and then halts.

In particular, simulation of piriform gives rise to temporal sequences of spatial patterns of synchronized cell firing in layer II, and the patterns change over time: the physiology and anatomy of the structure cause successive 'sniffs' of the same olfactory stimulus to give rise to a sequence of spatial patterns, each of which encodes successively more specific information about the stimulus, beginning with its similarity to other previously-learned stimuli, and ending with a unique encoding of its characteristics. It is possible that both the early similarity-based 'cluster' information and the late unique encodings are used, for different purposes, by brain structures that receive these signals as output from piriform.

## ACKNOWLEDGEMENTS

Much of the theoretical underpinning of this work depends critically on data generated by John Larson; we are grateful for his insightful advice and help. This work has benefited from discussions with Michel Baudry, Mark Gluck, and Ursula Staubli. José Ambros-Ingerson is supported by a fellowship from Hewlett-Packard, México, administered by UC MEXUS.

## Footnotes

[1]This research was supported in part by the Office of Naval Research under grants N00014-84-K-0391 and N00014-87-K-0838 and by the National Science Foundation under grant IST-85-12419.

## REFERENCES

Bliss, T.V.P. and Lømo, T. (1973). Long-lasting potentiation of synaptic transmission in the dentate area of the anesthetized rabbit following stimulation of the perforant path. *J.Physiol.Lond.* 232:357–374.

Feldman, J.A. (1982). Dynamic connections in neural networks. *Biological Cybernetics* 46:27–39.

Haberly, L.B. (1985). Neuronal circuitry in olfactory cortex: Anatomy and functional implications. *Chemical Senses* 10:219–238.

Haberly, L.B. and J.L. Price (1977). The axonal projection patterns of the mitral and tufted cells of the olfactory bulb in the rat. *Brain Res* 129:152–157.

Haberly, L.B. and J.L. Price (1978a). Association and commissural fiber systems of the olfactory cortex of the rat. I. Systems originating in the piriform cortex and adjacent areas. *J. Comp. Neurol.* 178:711–740.

Haberly, L.B. and J.L. Price (1978b). Association and commissural fiber systems of the olfactory cortex of the rat. II. Systems originating in the olfactory peduncle. *J. Comp. Neurol.* 181:781–808.

Hebb, D.O. (1949). *The Organization of Behavior*. New York: Wiley.

Krettek, J.E. and J.L. Price (1977). Projections from the amygdaloid complex and adjacent olfactory structures to the entorhinal cortex and to the subiculum in the rat and cat. *J Comp Neurol* 172:723–752.

Larson, J. and G. Lynch (1986). Synaptic potentiation in hippocampus by patterned stimulation involves two events. *Science* 232:985–988.

Lee, K., Schottler, F., Oliver, M. and Lynch, G. (1980). Brief bursts of high-frequency stimulation produce two types of structural change in rat hippocampus. *J.Neurophysiol.* 44:247–258.

Lynch, G. and Baudry, M. (1984). The biochemistry of memory: a new and specific hypothesis. *Science* 224:1057–1063.

Lynch, G. (1986). Synapses, circuits, and the beginnings of memory. Cambridge, Mass: MIT Press.

Lynch, G., Larson, J., Staubli, U., and Baudry, M. (1987). New perspectives on the physiology, chemistry and pharmacology of memory. *Drug Devel.Res.* 10:295–315.

Lynch, G., Granger, R., Levy, W. and Larson, J. (1988). Some possible functions of simple cortical networks suggested by computer modeling. In: *Neural Models of Plasticity: Theoretical and Empirical Approaches*, Byrne, J. and Berry, W.O. (Eds.), (in press).

Lynch, G. and Granger, R. (1988). Simulation and analysis of a cortical network. *The Psychology of Learning and Motivation*, Vol.22 (in press).

Luskin, M.B. and J.L. Price (1983). The laminar distribution of intracortical fibers orginating in the olfactory cortex of the rat. *J Comp Neurol* 216:292–302.

Parker, D.B. (1985). Learning-logic. MIT TR-47, Massachusetts Institute of Technology, Center for Computational Research in Economics and Management Science, Cambridge, Mass.

Price, J.L. (1973). An autoradiographic study of complementary laminar patterns of termination of afferent fibers to the olfactory cortex. *J.Comp.Neur.* 150:87–108.

Price, J.L. and B.M. Slotnick (1983). Dual olfactory representation in the rat thalamus: An anatomical and electrophysiological study. *J Comp Neurol* 215:63–77.

Roman, F., Staubli, U. and Lynch, G. (1987). Evidence for synaptic potentiation in a cortical network during learning. *Brain Res.* 418:221–226.

Rosenblatt, F. (1962). Principles of neurodynamics. New York: Spartan.

Rumelhart, D., Hinton, G. and Williams, R. (1986). Learning Internal Representations by Error Propagation. In D.Rumelhart and J.McClelland (Eds.), *Parallel*

*Distributed Processing*, Cambridge: MIT Press.

Sivilotti, M.A., Mahowald, M.A. and Mead, C.A. (1987). Real-time visual computations using analog CMOS processing arrays. In: Advanced Research in VLSI (Ed. Paul Losleben), MIT Press, Cambridge.

Staubli, U. and Lynch, G. (1987). Stable hippocampal long-term potentiation elicited by "theta" pattern stimulation. *Brain Res.* (in press).

Widrow, G. and Hoff, M.E. (1960). Adaptive Switching Circuits. *Institute of Radio Engineers, Western Electronic Convention Record, Part 4*, pp.96–104.

Wigstrøm, H., B. Gustaffson, Y.Y. Huang and W.C. Abraham (1986). Hippocampal long-term potentiation is induced by pairing single afferent volleys with intracellularly injected depolarizing current pulses. *Acta Physiol Scand* 126:317–319.

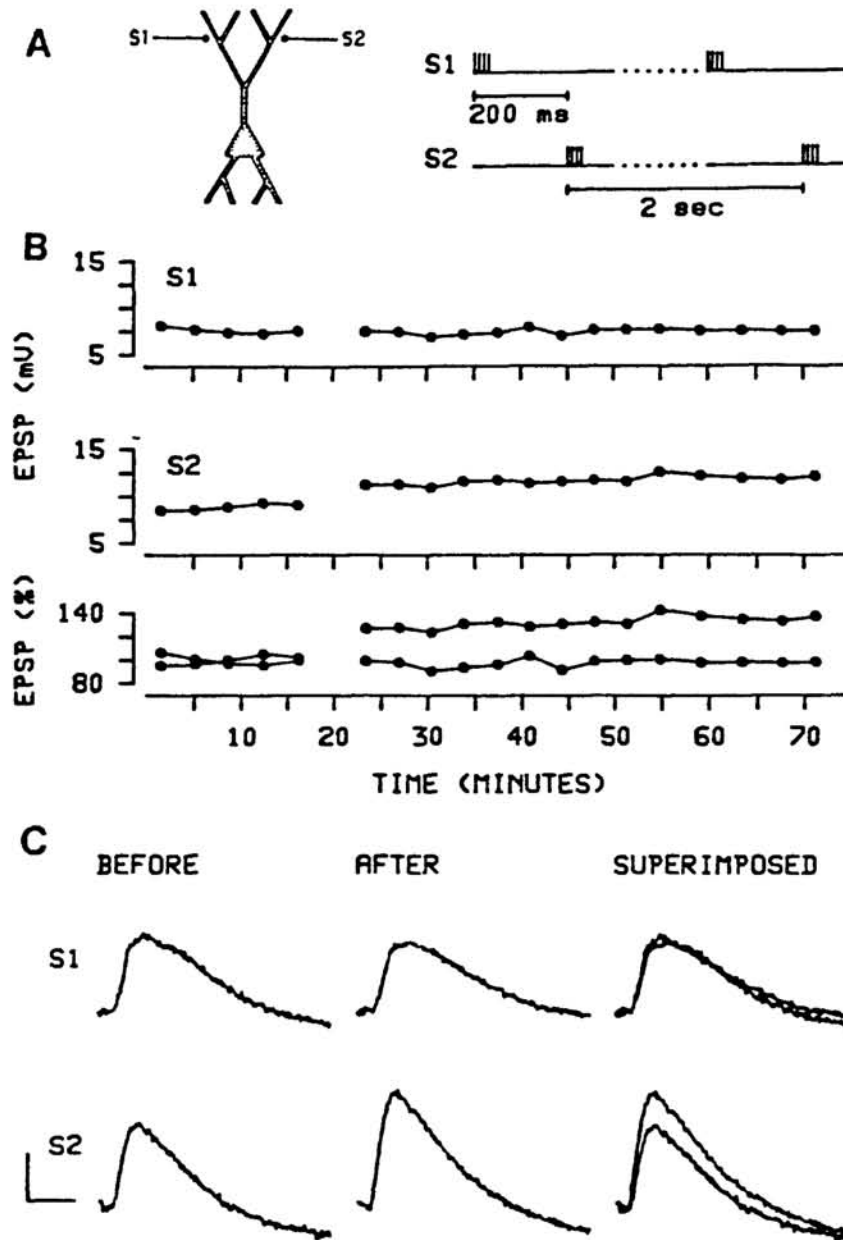

**Figure 1.** LTP induction by short high-frequency bursts involves sequential "priming" and "consolidation" events.

A) S1 and S2 represent separate groups of Shaffer/commissural fibers converging on a single CA1 pyramidal neuron. The stimulation pattern employed consisted of pairs of bursts (each 4 pulses at 100Hz) given to S1 and S2 respectively, with a 200ms delay between them. The pairs were repeated 10 times at 2 sec intervals.

B) Only the synapses activated by the delayed burst (S2) showed LTP. The top panel shows measurements of amplitudes of intracellular EPSPs evoked by single pulses to S1 before and after patterned stimulation (given at 20 min into the experiment). The middle panel shows the amplitude of EPSPs evoked by S2. Bottom panel shows EPSP amplitudes for both pathways expressed as a percentage of their respective sizes before burst stimulation.

C) Shown are records of EPSPs evoked by S1 and S2 five min. before and 40 min. after patterned burst stimulation. Calibration bar: 5mV, 5msec. (From Larson and Lynch, 1986).

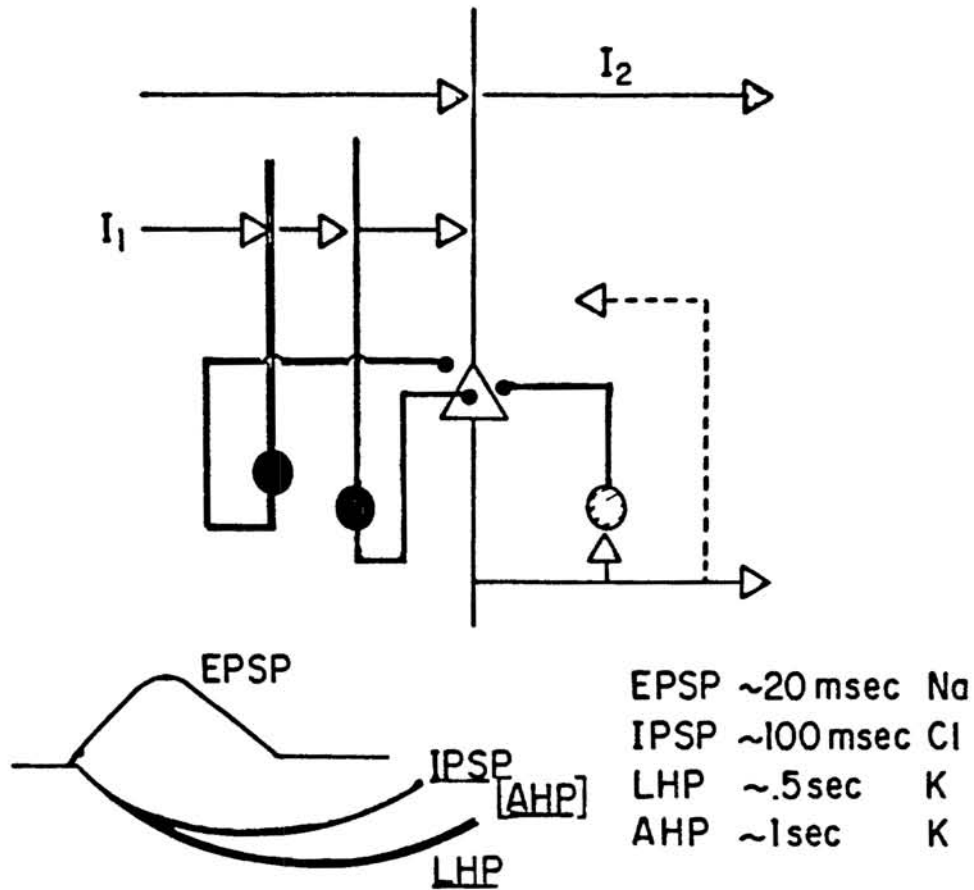

EPSP

IPSP
[AHP]

LHP

| EPSP | ~20 msec | Na |
| IPSP | ~100 msec | Cl |
| LHP | ~.5 sec | K |
| AHP | ~1 sec | K |

**Figure 2.** Onset and duration of events comprising stimulation of a layer II cell in piriform cortex. Axonal stimulation via the lateral olfactory tract (LOT) activates feedforward EPSPs with rapid onset and short duration ($\approx$20msec) and two types of feedforward inhibition: short feedforward IPSPs with slower onset and somewhat longer duration ($\approx$100msec) than the EPSPs, and longer hyperpolarizing potentials (LHP) lasting $\approx$500msec. These two types of inhibition are not specific to firing cells; an additional, very long-lasting ($\approx$1sec) inhibitory afterhyperpolarizing current (AHP) is induced in a cell-specific fashion in those cells with intense firing activity. Finally, feedback EPSPs and IPSPs are induced by activation via recurrent collateral axons from layer II cells.

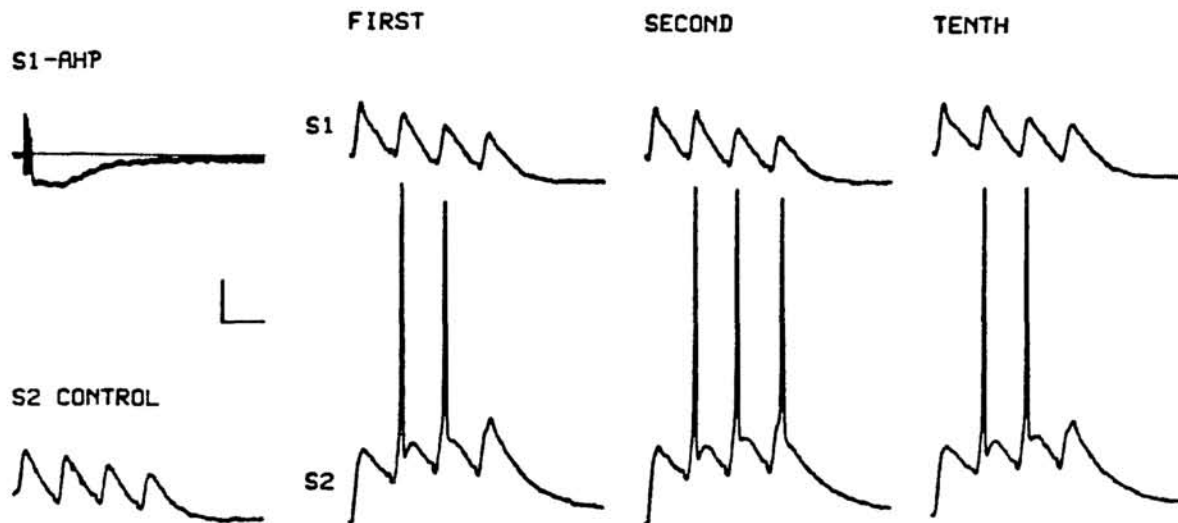

**Figure 3.** When short, high-frequency bursts are input to cells 200ms after an initial 'priming' event, the broadened EPSPs (see Figure 1) will allow the contributions of the second and subsequent pulses comprising the burst to sum with the depolarization of the first pulse, yielding higher postsynaptic depolarization sufficient to cause the cell to spike. (From Lynch, Larson, Staubli and Baudry, 1987).

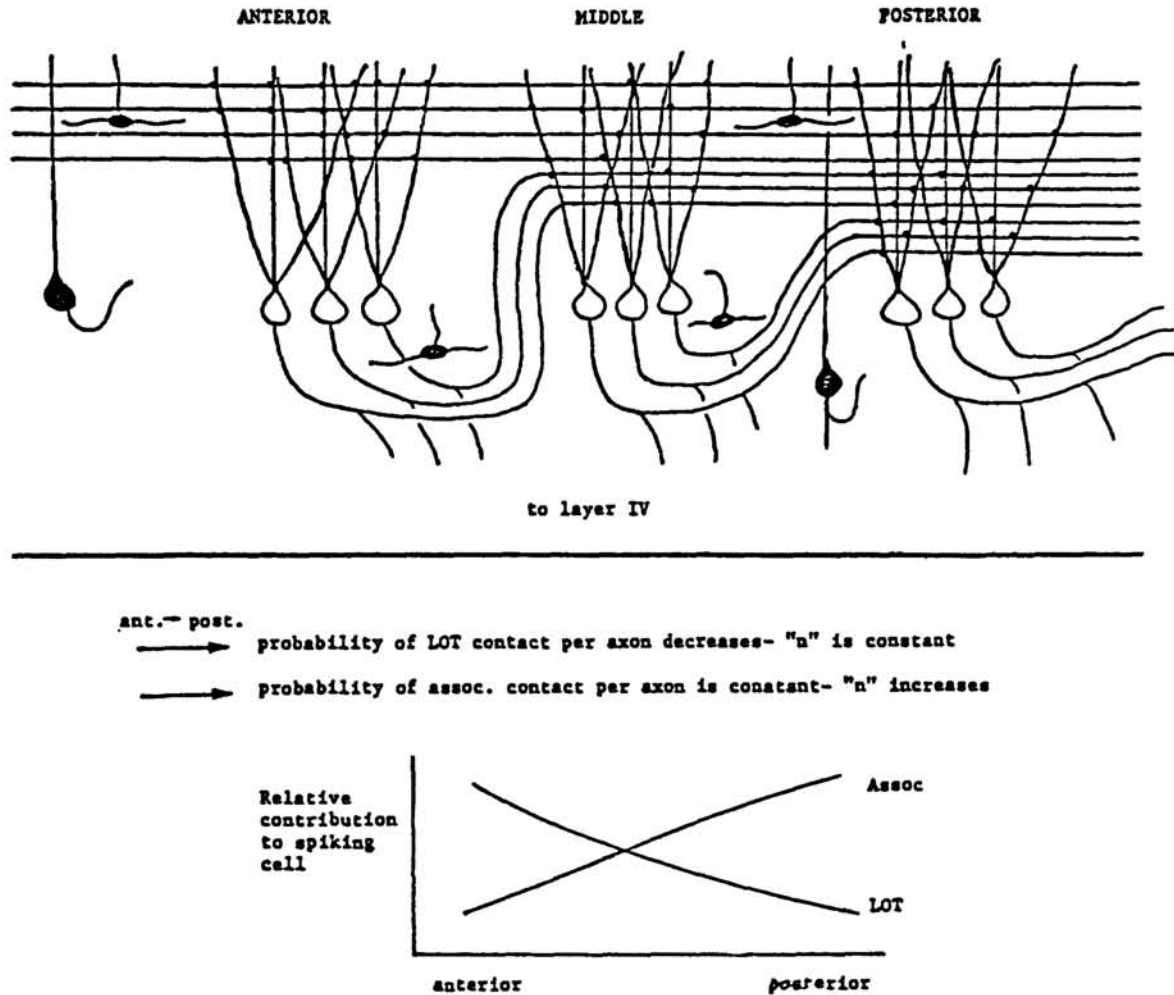

Figure 4. Organization of extrinsic and feedback inputs to layer-II cells of piriform cortex. The axons comprising the lateral olfactory tract (LOT), originating from the bulb, innervate distal dendrites, whereas the feedback collateral or associational fibers contact proximal dendrites. Layer II cells in anterior (rostral) piriform are depicted as being dominated by extrinsic (LOT) input, whereas feedback inputs are more prominent on cells in posterior (caudal) piriform.

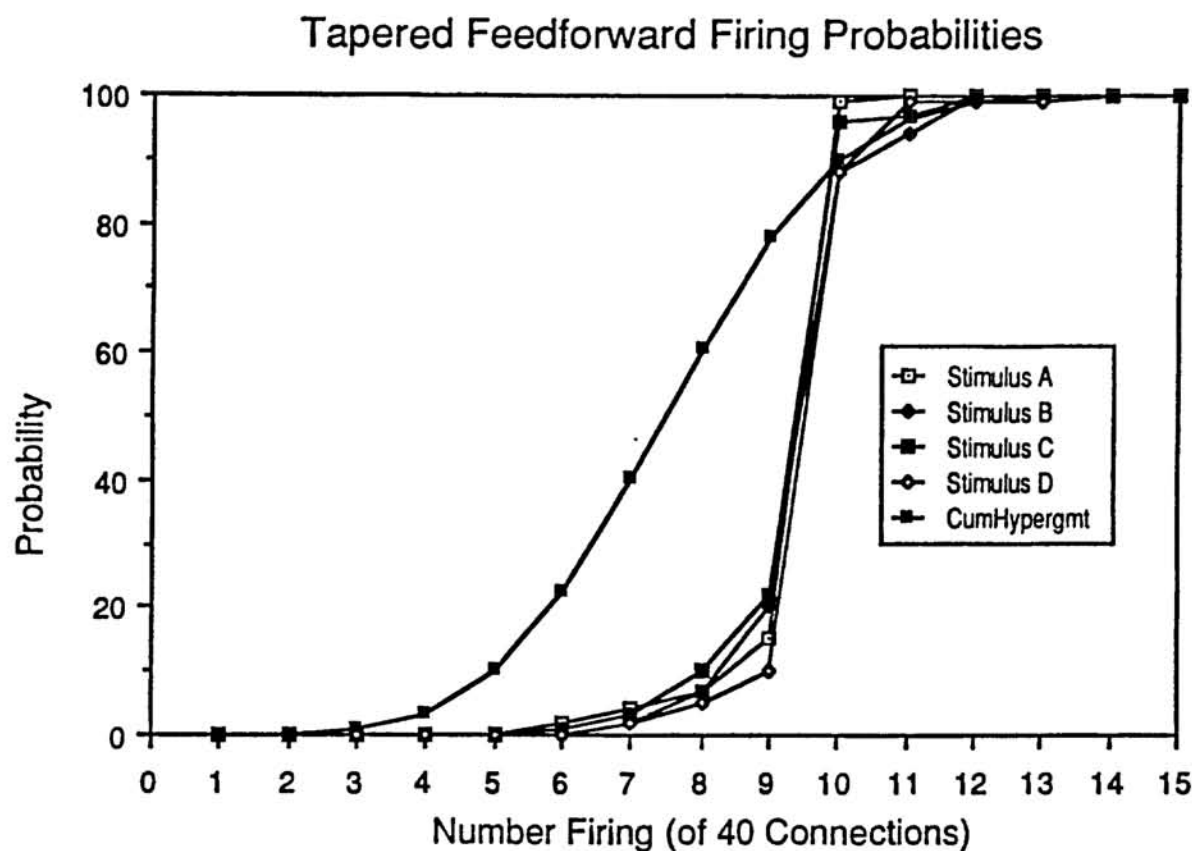

**Figure 5.** Probability of layer-II-cell firing as a function of number of LOT axons active, in the absence of local inhibitory patches. The hypergeometric function ('CumHypergmt') specifies the probability of layer II cell firing in the absence of caudally-directed feedback collaterals, i.e., assuming that all collaterals are equally probable to travel either rostrally or caudally. In this case, there is a smooth S-shaped function for probability of cell firing with increasing LOT activity, so that adjustment of global firing threshold (e.g., via nonspecific cholinergic inputs affecting all piriform inhibitory interneurons) can effectively normalize piriform layer II cell firing. However, when feedback axons are caudally directed, then probability steepens markedly, becoming a near step function, in which the probability of cell firing is exquisitely sensitive to the number of active inputs, across a range of empirically-tested LOT stimulation patterns (A – D in the figure). In this case, global adjustment of inhibition will fail to adequately normalize layer II cell firing: the probability of cell firing will always be either near zero or near 1.0; i.e., either nearly all cells will fire or almost none will fire. Local inhibitory control of 'patches' of layer II solve this problem (refer to text).